# A Scalable Hierarchical Distributed Language Model

**Andriy Mnih**
Department of Computer Science
University of Toronto
amnih@cs.toronto.edu

**Geoffrey Hinton**
Department of Computer Science
University of Toronto
hinton@cs.toronto.edu

## Abstract

Neural probabilistic language models (NPLMs) have been shown to be competitive with and occasionally superior to the widely-used $n$-gram language models. The main drawback of NPLMs is their extremely long training and testing times. Morin and Bengio have proposed a hierarchical language model built around a binary tree of words, which was two orders of magnitude faster than the non-hierarchical model it was based on. However, it performed considerably worse than its non-hierarchical counterpart in spite of using a word tree created using expert knowledge. We introduce a fast hierarchical language model along with a simple feature-based algorithm for automatic construction of word trees from the data. We then show that the resulting models can outperform non-hierarchical neural models as well as the best $n$-gram models.

## 1 Introduction

Statistical language modelling is concerned with building probabilistic models of word sequences. Such models can be used to discriminate probable sequences from improbable ones, a task important for performing speech recognition, information retrieval, and machine translation. The vast majority of statistical language models are based on the Markov assumption, which states that the distribution of a word depends only on some fixed number of words that immediately precede it. While this assumption is clearly false, it is very convenient because it reduces the problem of modelling the probability distribution of word sequences of arbitrary length to the problem of modelling the distribution on the next word given some fixed number of preceding words, called the context. We will denote this distribution by $P(w_n|w_{1:n-1})$, where $w_n$ is the next word and $w_{1:n-1}$ is the context $(w_1, ..., w_{n-1})$.

$n$-gram language models are the most popular statistical language models due to their simplicity and surprisingly good performance. These models are simply conditional probability tables for $P(w_n|w_{1:n-1})$, estimated by counting the $n$-tuples in the training data and normalizing the counts appropriately. Since the number of $n$-tuples is exponential in $n$, smoothing the raw counts is essential for achieving good performance. There is a large number of smoothing methods available for $n$-gram models [4]. In spite of the sophisticated smoothing methods developed for them, $n$-gram models are unable to take advantage of large contexts since the data sparsity problem becomes extreme. The main reason for this behavior is the fact that classical $n$-gram models are essentially conditional probability tables where different entries are estimated independently of each other. These models do not take advantage of the fact that similar words occur in similar contexts, because they have no concept of similarity. Class-based $n$-gram models [3] aim to address this issue by clustering words and/or contexts into classes based on their usage patterns and then using this class information to improve generalization. While it can improve $n$-gram performance, this approach introduces a very rigid kind of similarity, since each word typically belongs to exactly one class.

An alternative and much more flexible approach to counteracting the data sparsity problem is to represent each word using a real-valued feature vector that captures its properties, so that words

used in similar contexts will have similar feature vectors. Then the conditional probability of the next word can be modelled as a smooth function of the feature vectors of the context words and the next word. This approach provides automatic smoothing, since for a given context similar words are now guaranteed to be assigned similar probabilities. Similarly, similar contexts are now likely to have similar representations resulting in similar predictions for the next word. Most models based on this approach use a feed-forward neural network to map the feature vectors of the context words to the distribution for the next word (e.g. [12], [5], [9]). Perhaps the best known model of this type is the Neural Probabilistic Language Model [1], which has been shown to outperform $n$-gram models on a dataset of about one million words.

## 2   The hierarchical neural network language model

The main drawback of the NPLM and other similar models is that they are very slow to train and test [10]. Since computing the probability of the next word requires explicitly normalizing over all words in the vocabulary, the cost of computing the probability of the given next word and the cost of computing the full distribution over the next word are virtually the same – they take time linear in the vocabulary size. Since computing the exact gradient in such models requires repeatedly computing the probability of the next word given its context and updating the model parameters to increase that probability, training time is also linear in the vocabulary size. Typical natural language datasets have vocabularies containing tens of thousands of words, which means that training NPLM-like models the straightforward way is usually too computationally expensive in practice. One way to speed up the process is to use a specialized importance sampling procedure to approximate the gradients required for learning [2]. However, while this method can speed up training substantially, testing remains computationally expensive.

The hierarchical NPLM introduced in [10], provides an exponential reduction in time complexity of learning and testing as compared to the NPLM. It achieves this reduction by replacing the unstructured vocabulary of the NPLM by a binary tree that represents a hierarchical clustering of words in the vocabulary. Each word corresponds to a leaf in the tree and can be uniquely specified by the path from the root to that leaf. If $N$ is the number of words in the vocabulary and the tree is balanced, any word can be specified by a sequence of $O(\log N)$ binary decisions indicating which of the two children of the current node is to be visited next. This setup replaces one $N$-way choice by a sequence of $O(\log N)$ binary choices. In probabilistic terms, one $N$-way normalization is replaced by a sequence of $O(\log N)$ local (binary) normalizations. As a result, a distribution over words in the vocabulary can be specified by providing the probability of visiting the left child at each of the nodes. In the hierarchical NPLM, these local probabilities are computed by giving a version of the NPLM the feature vectors for the context words as well as a feature vector for the current node as inputs. The probability of the next word is then given by the probability of making a sequence of binary decisions that corresponds to the path to that word.

When applied to a dataset of about one million words, this model outperformed class-based trigrams, but performed considerably worse than the NPLM [10]. The hierarchical model however was more than two orders of magnitude faster than the NPLM. The main limitation of this work was the procedure used to construct the tree of words for the model. The tree was obtained by starting with the WordNet IS-A taxonomy and converting it into a binary tree through a combination of manual and data-driven processing. Our goal is to replace this procedure by an automated method for building trees from the training data without requiring expert knowledge of any kind. We will also explore the performance benefits of using trees where each word can occur more than once.

## 3   The log-bilinear model

We will use the log-bilinear language model (LBL) [9] as the foundation of our hierarchical model because of its excellent performance and simplicity. Like virtually all neural language models, the LBL model represents each word with a real-valued feature vector. We will denote the feature vector for word $w$ by $r_w$ and refer to the matrix containing all these feature vectors as $R$. To predict the next word $w_n$ given the context $w_{1:n-1}$, the model computes the predicted feature vector $\hat{r}$ for the next word by linearly combining the context word feature vectors:

$$\hat{r} = \sum_{i=1}^{n-1} C_i r_{w_i}, \tag{1}$$

where $C_i$ is the weight matrix associated with the context position $i$. Then the similarity between the predicted feature vector and the feature vector for each word in the vocabulary is computed using the inner product. The similarities are then exponentiated and normalized to obtain the distribution over the next word:

$$P(w_n = w|w_{1:n-1}) = \frac{\exp(\hat{r}^T r_w + b_w)}{\sum_j \exp(\hat{r}^T r_j + b_j)}. \tag{2}$$

Here $b_w$ is the bias for word $w$, which is used to capture the context-independent word frequency.

Note that the LBL model can be interpreted as a special kind of a feed-forward neural network with one linear hidden layer and a softmax output layer. The inputs to the network are the feature vectors for the context words, while the matrix of weights from the hidden layer to the output layer is simply the feature vector matrix $R$. The vector of activities of the hidden units corresponds to the the predicted feature vector for the next word. Unlike the NPLM, the LBL model needs to compute the hidden activities only once per prediction and has no nonlinearities in its hidden layer. In spite of its simplicity the LBL model performs very well, outperforming both the NPLM and the $n$-gram models on a fairly large dataset [9].

## 4  The hierarchical log-bilinear model

Our hierarchical language model is based on the hierarchical model from [10]. The distinguishing features of our model are the use of the log-bilinear language model for computing the probabilities at each node and the ability to handle multiple occurrences of each word in the tree. Note that the idea of using multiple word occurrences in a tree was proposed in [10], but it was not implemented.

The first component of the hierarchical log-bilinear model (HLBL) is a binary tree with words at its leaves. For now, we will assume that each word in the vocabulary is at exactly one leaf. Then each word can be uniquely specified by a path from the root of the tree to the leaf node the word is at. The path itself can be encoded as a binary string $d$ of decisions made at each node, so that $d_i = 1$ corresponds to the decision to visit the left child of the current node. For example, the string "10" corresponds to a path that starts at the root, visits its left child, and then visits the right child of that child. This allows each word to be represented by a binary string which we will call a code.

The second component of the HLBL model is the probabilistic model for making the decisions at each node, which in our case is a modified version of the LBL model. In the HLBL model, just like in its non-hierarchical counterpart, context words are represented using real-valued feature vectors. Each of the non-leaf nodes in the tree also has a feature vector associated with it that is used for discriminating the words in the left subtree form the words in the right subtree of the node. Unlike the context words, the words being predicted are represented using their binary codes that are determined by the word tree. However, this representation is still quite flexible, since each binary digit in the code encodes a decision made at a node, which depends on that node's feature vector.

In the HLBL model, the probability of the next word being $w$ is the probability of making the sequences of binary decisions specified by the word's code, given the context. Since the probability of making a decision at a node depends only on the predicted feature vector, determined by the context, and the feature vector for that node, we can express the probability of the next word as a product of probabilities of the binary decisions:

$$P(w_n = w|w_{1:n-1}) = \prod_i P(d_i|q_i, w_{1:n-1}), \tag{3}$$

where $d_i$ is $i^{th}$ digit in the code for word $w$, and $q_i$ is the feature vector for the $i^{th}$ node in the path corresponding to that code. The probability of each decision is given by

$$P(d_i = 1|q_i, w_{1:n-1}) = \sigma(\hat{r}^T q_i + b_i), \tag{4}$$

where $\sigma(x)$ is the logistic function and $\hat{r}$ is the predicted feature vector computed using Eq. 1. $b_i$ in the equation is the node's bias that captures the context-independent tendency to visit the left child when leaving this node.

The definition of $P(w_n = w|w_{1:n-1})$ can be extended to multiple codes per word by including a summation over all codes for $w$ as follows:

$$P(w_n = w|w_{1:n-1}) = \sum_{d \in D(w)} \prod_i P(d_i|q_i, w_{1:n-1}),\qquad(5)$$

where $D(w)$ is a set of codes corresponding to word $w$. Allowing multiple codes per word can allow better prediction of words that have multiple senses or multiple usage patterns. Using multiple codes per word also makes it easy to combine several separate words hierarchies to into a single one to to reflect the fact that no single hierarchy can express all the relationships between words.

Using the LBL model instead of the NPLM for computing the local probabilities allows us to avoid computing the nonlinearities in the hidden layer which makes our hierarchical model faster at making predictions than the hierarchical NPLM. More importantly, the hierarchical NPLM needs to compute the hidden activities once for each of the $O(\log N)$ decisions, while the HLBL model computes the predicted feature vector just once per prediction. However, the time complexity of computing the probability for a single binary decision in an LBL model is still quadratic in the feature vector dimensionality $D$, which might make the use of high-dimensional feature vectors too computationally expensive. We make the time complexity linear in $D$ by restricting the weight matrices $C_i$ to be diagonal.[1] Note that for a context of size 1, this restriction does not reduce the representational power of the model because the context weight matrix $C_1$ can be absorbed into the word feature vectors. And while this restriction does makes the models with larger contexts slightly less powerful, we believe that this loss is more than compensated for by much faster training times which allow using more complex trees.

HLBL models can be trained by maximizing the (penalized) log-likelihood. Since the probability of the next word depends only on the context weights, the feature vectors of the context words, and the feature vectors of the nodes on the paths from the root to the leaves containing the word in question, only a (logarithmically) small fraction of the parameters need to be updated for each training case.

## 5   Hierarchical clustering of words

The first step in training a hierarchical language model is constructing a binary tree of words for the model to use. This can be done by using expert knowledge, data-driven methods, or a combination of the two. For example, in [10] the tree was constructed from the IS-A taxonomy DAG from WordNet [6]. After preprocessing the taxonomy by hand to ensure that each node had only one parent, data-driven hierarchical binary clustering was performed on the children of the nodes in the taxonomy that had more than two children, resulting in a binary tree.

We are interested in using a pure learning approach applicable in situations where the expert knowledge is unavailable. It is also not clear that using expert knowledge, even when it is available, will lead to superior performance. Hierarchical binary clustering of words based on the their usage statistics is a natural choice for generating binary trees of words automatically. This task is similar to the task of clustering words into classes for training class-based $n$-gram models, for which a large number of algorithms has been proposed. We considered several of these algorithms before deciding to use our own algorithm which turned out to be surprisingly effective in spite of its simplicity. However, we will mention two existing algorithms that might be suitable for producing binary word hierarchies. Since we wanted an algorithm that scaled well to large vocabularies, we restricted our attention to the top-down hierarchical clustering algorithms, as they tend to scale better than their agglomerative counterparts [7]. The algorithm from [8] produces exactly the kind of binary trees we need, except that its time complexity is cubic in the vocabulary size.[2] We also considered the distributional clustering algorithm [11] but decided not to use it because of the difficulties involved in using contexts of more than one word for clustering. This problem is shared by most $n$-gram clustering algorithms, so we will describe it in some detail. Since we would like to cluster words for easy prediction of the next word based on its context, it is natural to describe each word in terms of the contexts that can precede it. For example, for a single-word context one such description is the

distribution of words that precede the word of interest in the training data. The problem becomes apparent when we consider using larger contexts: the number of contexts that can potentially precede a word grows exponentially in the context size. This is the very same data sparsity problem that affects the $n$-gram models, which is not surprising, since we are trying to describe words in terms of exponentially large (normalized) count vectors. Thus, clustering words based on such large-context representations becomes non-trivial due to the computational cost involved as well as the statistical difficulties caused by the sparsity of the data.

We avoid these difficulties by operating on low-dimensional real-valued word representations in our tree-building procedure. Since we need to train a model to obtain word feature vectors, we perform the following bootstrapping procedure: we generate a *random* binary tree of words, train an HLBL model based on it, and use the distributed representations it learns to represent words when building the word tree.

Since each word is represented by a distribution over contexts it appears in, we need a way of compressing such a collection of contexts down to a low-dimensional vector. After training the HLBL model, we summarize each context $w_{1:n-1}$ with the predicted feature vector produced from it using Eq. 1. Then, we condense the distribution of contexts that precede a given word into a feature vector by computing the expectation of the predicted representation w.r.t. that distribution. Thus, for the purposes of clustering each word is represented by its average predicted feature vector. After computing the low-dimensional real-valued feature vectors for words, we recursively apply a very simple clustering algorithm to them. At each step, we fit a mixture of two Gaussians to the feature vectors and then partition them into two subsets based on the responsibilities of the two mixture components for them. We then partition each of the subsets using the same procedure, and so on. The recursion stops when the current set contains only two words. We fit the mixtures by running the EM algorithm for 10 steps[3]. The algorithm updates both the means and the spherical covariances of the components. Since the means of the components are initialized based on a random partitioning of the feature vectors, the algorithm is not deterministic and will produce somewhat different clusterings on different runs. One appealing property of this algorithm is that the running time of each iteration is linear in the vocabulary size, which is a consequence of representing words using feature vectors of fixed dimensionality. In our experiments, the algorithm took only a few minutes to build a hierarchy for a vocabulary of nearly 18000 words based on 100-dimensional feature vectors.

The goal of an algorithm for generating trees for hierarchical language models is to produce trees that are well-supported by the data and are reasonably well-balanced so that the resulting models generalize well and are fast to train and test. To explore the trade-off between these two requirements, we tried several splitting rules in our tree-building algorithm. The rules are based on the observation that the responsibility of a component for a datapoint can be used as a measure of confidence about the assignment of the datapoint to the component. Thus, when the responsibilities of both components for a datapoint are close to 0.5, we cannot be sure that the datapoint should be in one component but not the other.

Our simplest rule aims to produce a balanced tree at any cost. It sorts the responsibilities and splits the words into two disjoint subsets of equal size based on the sorted order. The second rule makes splits well-supported by the data even if that results in an unbalanced tree. It achieves that by assigning the word to the component with the higher responsibility for the word. The third and the most sophisticated rule is an extension of the second rule, modified to assign a point to both components whenever both responsibilities are within $\epsilon$ of 0.5, for some pre-specified $\epsilon$. This rule is designed to produce multiple codes for words that are difficult to cluster. We will refer to the algorithms that use these rules as BALANCED, ADAPTIVE, and ADAPTIVE($\epsilon$) respectively. Finally, as a baseline for comparison with the above algorithms, we will use an algorithm that generates random balanced trees. It starts with a random permutation of the words and recursively builds the left subtree based one the first half of the words and the right subtree based on the second half of the words. We will call this algorithm RANDOM.

Table 1: Trees of words generated by the feature-based algorithm. The mean code length is the sum of lengths of codes associated with a word, averaged over the distribution of the words in the training data. The run-time complexity of the hierarchical model is linear in the mean code length of the tree used. The mean number of codes per word refers to the number of codes per word averaged over the training data distribution. Since each non-leaf node in a tree has its own feature vector, the number of free parameters associated with the tree is linear in this quantity.

| Tree label | Generating algorithm | Mean code length | Mean number of codes per word | Number of non-leaf nodes |
|---|---|---|---|---|
| T1 | RANDOM | 14.2 | 1.0 | 17963 |
| T2 | BALANCED | 14.3 | 1.0 | 17963 |
| T3 | ADAPTIVE | 16.1 | 1.0 | 17963 |
| T4 | ADAPTIVE(0.25) | 24.2 | 1.3 | 22995 |
| T5 | ADAPTIVE(0.4) | 29.0 | 1.7 | 30296 |
| T6 | ADAPTIVE(0.4) $\times$ 2 | 69.1 | 3.4 | 61014 |
| T7 | ADAPTIVE(0.4) $\times$ 4 | 143.2 | 6.8 | 121980 |

Table 2: The effect of the feature dimensionality and the word tree used on the test set perplexity of the model.

| Feature dimensionality | Perplexity using a random tree | Perplexity using a non-random tree | Reduction in perplexity |
|---|---|---|---|
| 25 | 191.6 | 162.4 | 29.2 |
| 50 | 166.4 | 141.7 | 24.7 |
| 75 | 156.4 | 134.8 | 21.6 |
| 100 | 151.2 | 131.3 | 19.9 |

# 6  Experimental results

We compared the performance of our models on the APNews dataset containing the Associated Press news stories from 1995 and 1996. The dataset consists of a 14 million word training set, a 1 million word validation set, and 1 million word test set. The vocabulary size for this dataset is 17964. We chose this dataset because it had already been used to compare the performance of neural models to that of $n$-gram models in [1] and [9], which allowed us to compare our results to the results in those papers. Except for where stated otherwise, the models used for the experiments used 100 dimensional feature vectors and a context size of 5. The details of the training procedure we used are given in the appendix. All models were compared based on their perplexity score on the test set.

We started by training a model that used a tree generated by the RANDOM algorithm (tree T1 in Table 1). The feature vectors learned by this model were used to build a tree using the BALANCED algorithm (tree T2). We then trained models of various feature vector dimensionality on each of these trees to see whether a highly expressive model can compensate for using a poorly constructed tree. The test scores for the resulting models are given in Table 2. As can be seen from the scores, using a non-random tree results in much better model performance. Though the gap in performance can be reduced by increasing the dimensionality of feature vectors, using a non-random tree drastically improves performance even for the model with 100-dimensional feature vectors. It should be noted however, that models that use the random tree are not entirely hopeless. For example, they outperform the unigram model which achieved the perplexity of 602.0 by a very large margin. This suggests that the HLBL architecture is sufficiently flexible to make effective use of a random tree over words.

Since increasing the feature dimensionality beyond 100 did not result in a substantial reduction in perplexity, we used 100-dimensional feature vectors for all of our models in the following experiments. Next we explored the effect of the tree building algorithm on the performance of the resulting HLBL model. To do that, we used the RANDOM, BALANCED, and ADAPTIVE algorithms to generate one tree each. The ADAPTIVE($\epsilon$) algorithm was used to generate two trees: one with $\epsilon$ set

Table 3: Test set perplexity results for the hierarchical LBL models. All the distributed models in the comparison used 100-dimensional feature vectors and a context size of 5. LBL is the non-hierarchical log-bilinear model. KN$n$ is a Kneser-Ney $n$-gram model. The scores for LBL, KN3, and KN5 are from [9]. The timing for LBL is based on our implementation of the model.

| Model type | Tree used | Tree generating algorithm | Perplexity | Minutes per epoch |
|---|---|---|---|---|
| HLBL | T1 | RANDOM | 151.2 | 4 |
| HLBL | T2 | BALANCED | 131.3 | 4 |
| HLBL | T3 | ADAPTIVE | 127.0 | 4 |
| HLBL | T4 | ADAPTIVE(0.25) | 124.4 | 6 |
| HLBL | T5 | ADAPTIVE(0.4) | 123.3 | 7 |
| HLBL | T6 | ADAPTIVE(0.4) $\times$ 2 | 115.7 | 16 |
| HLBL | T7 | ADAPTIVE(0.4) $\times$ 4 | 112.1 | 32 |
| LBL | – | – | 117.0 | 6420 |
| KN3 | – | – | 129.8 | – |
| KN5 | – | – | 123.2 | – |

to 0.25 and the other with $\epsilon$ set to 0.4. We then generated a $2\times$ overcomplete tree by running the ADAPTIVE($\epsilon = 0.4$) algorithm twice and creating a tree with a root node that had the two generated trees as its subtrees. Since the ADAPTIVE($\epsilon$) algorithm involves some randomization we tried to improve the model performance by allowing the model to choose dynamically between two possible clusterings. Finally, we generated a $4\times$ overcomplete using the same approach. Table 1 lists the generated trees as well as some statistics for them. Note that trees generated using ADAPTIVE($\epsilon$) using $\epsilon > 0$ result in models with more parameters due to the greater number of tree-nodes and thus tree-node feature vectors, as compared to trees generated using methods producing one code/leaf per word.

Table 3 shows the test set perplexities and time per epoch for the resulting models along with the perplexities for models from [9]. The results show that the performance of the HLBL models based on non-random trees is comparable to that of the $n$-gram models. As expected, building word trees adaptively improves model performance. The general trend that emerges is that bigger trees tend to lead to better performing models. For example, a model based on a single tree produced using the ADAPTIVE(0.4) algorithm, performs as well as the 5-gram but not as well as the non-hierarchical LBL model. However, using a $2\times$ overcomplete tree generated using the same algorithm results in a model that outperforms both the $n$-gram models and the LBL model, and using a $4\times$ overcomplete tree leads to a further reduction in perplexity. The time-per-epoch statistics reported for the neural models in Table 3 shows the great speed advantage of the HLBL models over the LBL model. Indeed, the slowest of our HLBL models is over 200 times faster than the LBL model.

## 7 Discussion and future work

We have demonstrated that a hierarchal neural language model can actually outperform its non-hierarchical counterparts and achieve state-of-the-art performance. The key to making a hierarchical model perform well is using a carefully constructed hierarchy over words. We have presented a simple and fast feature-based algorithm for automatic construction of such hierarchies. Creating hierarchies in which every word occurred more than once was essential to getting the models to perform better.

An inspection of trees generated by our adaptive algorithm showed that the words with the largest numbers of codes (i.e. the word that were replicated the most) were not the words with multiple distinct senses. Instead, the algorithm appeared to replicate the words that occurred relatively infrequently in the data and were therefore difficult to cluster. The failure to use multiple codes for words with several very different senses is probably a consequence of summarizing the distribution over contexts with a single mean feature vector when clustering words. The "sense multimodality" of context distributions would be better captured by using a small set of feature vectors found by clustering the contexts.

Finally, since our tree building algorithm is based on the feature vectors learned by the model, it is possible to periodically interrupt training of such a model to rebuild the word tree based on the feature vectors provided by the model being trained. This modified training procedure might produce better models by allowing the word hierarchy to adapt to the probabilistic component of the model and vice versa.

## Appendix: Details of the training procedure

The models have been trained by maximizing the log-likelihood using stochastic gradient ascent. All model parameters other than the biases were initialized by sampling from a Gaussian of small variance. The biases for the tree nodes were initialized so that the distribution produced by the model with all the non-bias parameters set to zero matched the base rates of the words in the training set.

Models were trained using the learning rate of $10^{-3}$ until the perplexity on the validation set started to increase. Then the learning rate was reduced to $3 \times 10^{-5}$ and training was resumed until the validation perplexity started increasing again. All model parameters were regulated using a small $L_2$ penalty.

## Acknowledgments

We thank Martin Szummer for his comments on a draft of this paper. This research was supported by NSERC and CFI. GEH is a fellow of the Canadian Institute for Advanced Research.

## Footnotes

[1]Thus the feature vector for the next word can now be computed as $\hat{r} = \sum_{i=1}^{n-1} c_i \circ r_{w_i}$, where $c_i$ is a vector of context weights for position $i$ and $\circ$ denotes the elementwise product of two vectors.

[2]More precisely, the time complexity of the algorithm is cubic in the number of the frequent words, but that is still to slow for our purposes.

[3]Running EM for more than 10 steps did not make a significant difference in the quality of the resulting trees.

## References

[1] Yoshua Bengio, Rejean Ducharme, Pascal Vincent, and Christian Jauvin. A neural probabilistic language model. *Journal of Machine Learning Research*, 3:1137–1155, 2003.

[2] Yoshua Bengio and Jean-Sébastien Senécal. Quick training of probabilistic neural nets by importance sampling. In *AISTATS'03*, 2003.

[3] P.F. Brown, R.L. Mercer, V.J. Della Pietra, and J.C. Lai. Class-based n-gram models of natural language. *Computational Linguistics*, 18(4):467–479, 1992.

[4] Stanley F. Chen and Joshua Goodman. An empirical study of smoothing techniques for language modeling. In *Proceedings of the Thirty-Fourth Annual Meeting of the Association for Computational Linguistics*, pages 310–318, San Francisco, 1996.

[5] Ahmad Emami, Peng Xu, and Frederick Jelinek. Using a connectionist model in a syntactical based language model. In *Proceedings of ICASSP*, volume 1, pages 372–375, 2003.

[6] C. Fellbaum et al. *WordNet: an electronic lexical database*. Cambridge, Mass: MIT Press, 1998.

[7] J. Goodman. A bit of progress in language modeling. Technical report, Microsoft Research, 2000.

[8] John G. McMahon and Francis J. Smith. Improving statistical language model performance with automatically generated word hierarchies. *Computational Linguistics*, 22(2):217–247, 1996.

[9] A. Mnih and G. Hinton. Three new graphical models for statistical language modelling. *Proceedings of the 24th international conference on Machine learning*, pages 641–648, 2007.

[10] Frederic Morin and Yoshua Bengio. Hierarchical probabilistic neural network language model. In Robert G. Cowell and Zoubin Ghahramani, editors, *AISTATS'05*, pages 246–252, 2005.

[11] F. Pereira, N. Tishby, and L. Lee. Distributional clustering of English words. *Proceedings of the 31st conference on Association for Computational Linguistics*, pages 183–190, 1993.

[12] Holger Schwenk and Jean-Luc Gauvain. Connectionist language modeling for large vocabulary continuous speech recognition. In *Proceedings of the International Conference on Acoustics, Speech and Signal Processing*, pages 765–768, 2002.

